# Recurrent Neural Networks Can Learn to Implement Symbol-Sensitive Counting

**Paul Rodriguez**
Department of Cognitive Science
University of California, San Diego
La Jolla, CA. 92093
prodrigu@cogsci.ucsd.edu

**Janet Wiles**
School of Information Technology and
Department of Psychology
University of Queensland
Brisbane, Queensland 4072 Australia
janetw@it.uq.edu.au

## Abstract

Recently researchers have derived formal complexity analysis of analog computation in the setting of discrete-time dynamical systems. As an empirical constrast, training recurrent neural networks (RNNs) produces self-organized systems that are realizations of analog mechanisms. Previous work showed that a RNN can learn to process a simple context-free language (CFL) by counting. Herein, we extend that work to show that a RNN can learn a harder CFL, a simple palindrome, by organizing its resources into a symbol-sensitive counting solution, and we provide a dynamical systems analysis which demonstrates how the network can not only count, but also copy and store counting information.

## 1 INTRODUCTION

Several researchers have recently derived results in analog computation theory in the setting of discrete-time dynamical systems(Siegelmann, 1994; Maass & Opren, 1997; Moore, 1996; Casey, 1996). For example, a dynamical recognizer (DR) is a discrete-time continuous dynamical system with a given initial starting point and a finite set of Boolean output decision functions(Pollack, 1991; Moore, 1996; see also Siegelmann, 1993). The dynamical system is composed of a space, $\Re^n$, an alphabet $A$, a set of functions (1 per element of $A$) that each maps $\Re^n \to \Re^n$ and an accepting region $H_{yes}$ in $\Re^n$. With enough precision and appropriate differential equations, DRs can use real-valued variables to encode contents of a stack or counter (for details see Siegelmann, 1994; Moore, 1996).

As an empirical contrast, training recurrent neural networks (RNNs) produces self-organized implementations of analog mechanisms. In previous work we showed that an RNN can learn to process a simple context-free language, $a^n b^n$, by organizing its resources into a counter which is similar to hand-coded dynamical recognizers but also exhibits some

novelties (Wiles & Elman, 1995). In particular, similar to hand-coded counters, the network developed proportional contracting and expanding rates and precision matters - but unexpectedly the network distributed the contraction/expansion axis among hidden units, developed a saddle point to transition between the first half and second half of a string, and used oscillating dynamics as a way to visit regions of the phase space around the fixed points. In this work we show that an RNN can implement a solution for a harder CFL, a simple palindrome language(described below), which requires a symbol-sensitive counting solution. We provide a dynamical systems analysis which demonstrates how the network can not only count, but also copy and store counting information implicitly in space around a fixed point.

## 2   TRAINING an RNN TO PROCESS CFLs

We use a discrete-time RNN that has 1 hidden layer with recurrent connections, and 1 output layer without recurrent connections so that the accepting regions are determined by the output units. The RNN processes output in Time($n$), where $n$ is the length of the input, and it can recognize languages that are a proper subset of context-sensitive languages and a proper superset of regular languages(Moore, 1996). Consequently, the RNN we investigate can in principle embody the computational power needed to process self-recursion.

Furthermore, many connectionist models of language processing have used a prediction task(e.g. Elman, 1990). Hence, we trained an RNN to be a real-time transducer version of a dynamical recognizer that predicts the next input in a sequence. Although the network does not explicitly accept or reject strings, if our network makes all the right predictions possible then performing the prediction task subsumes the accept task, and in principle one could simply reject unmatched predictions. We used a threshhold criterion of .5 such that if an ouput node has a value greater than .5 then the network is considered to be making that prediction. If the network makes all the right predictions possible for some input string, then it is correctly processing that string. Although a finite dimensional RNN cannot process CFLs robustly with a margin for error (e.g.Casey, 1996;Maass and Orponen,1997), we will show that it can acquire the right kind of trajectory to process the language in a way that generalizes to longer strings.

### 2.1   A SIMPLE PALINDROME LANGUAGE

A *palindrome* language (mirror language) consists of a set of strings, $S$, such that each string, $s \in S$, $s = ww^r$, is a concatenation of a substring, $w$, and its reverse, $w^r$. The relevant aspect of this language is that a mechanism cannot use a simple counter to process the string but must use the functional equivalent of a stack that enables it to match the symbols in second half of the string with the first half.

We investigated a palindrome language that uses only two symbols for $w$, two other symbols for $w^r$, such that the second half of the string is fully predictable once the change in symbols occurs. The language we used is a simple version restricted such that one symbol is always present and precedes the other, for example: $w = a^n b^m$, $w^r = B^m A^n$, e.g. $aaaabbbBBBAAAA$, (where $n > 0$, $m >= 0$). Note that the embedded subsequence $b^m B^m$ is just the simple-CFL used in Wiles & Elman (1995) as mentioned above, hence, one can reasonably expect that a solution to this task has an embedded counter for the subsequence $b...B$.

### 2.2   LINEAR SYSTEM COUNTERS

A basic counter in analog computation theory uses real-valued precision (e.g. Siegelman 1994; Moore 1996). For example, a 1-dimensional up/down counter for two symbols $\{a, b\}$

is the system $f(x) = .5x + .5a$, $f(x) = 2x - .5b$ where $x$ is the state variable, $a$ is the input variable to count up(push), and $b$ is the variable to count down(pop). A sequence of input *aaabbb* has state values(starting at 0): .5,.75,.875, .75,.5,0.

Similarly, for our transducer version one can develop piecewise linear system equations in which counting takes place along different dimensions so that different predictions can be made at appropriate time steps[1]. The linear system serves as a hypothesis before running any simulations to understand the implementation issues for an RNN. For example, using the function $f(x) = x$ for $x \in [0, 1]$, 0 for $x < 0$, 1 for $x > 1$, then for the simple palindrome task one can explicitly encode a mechanism to copy and store the count for $a$ across the $b...B$ subsequences. If we assign dimension-1 to $a$, dimension-2 to $b$, dimension-3 to $A$, dimension-4 to $B$, and dimension-5 to store the $a$ value, we can build a system so that for a sequence *aaabbBBAAA* we get state variables values: initial, (0,0,0,0,0), (.5,0,0,0,0), (.75,0,0,0,0), (.875,0,0,0,0), (0,.5,0,0,.875), (0,.75,0,0,.875), (0,0,0,.5,.875), (0,0,0,0,.875), (0,0,.75,0,0), (0,0,.5,0,0), (0,0,0,0,0). The matrix equations for such a system could be:

$$X_t = f(\begin{bmatrix} .5 & 0 & 0 & 0 & 0 \\ 0 & .5 & 0 & 0 & 0 \\ 0 & 0 & 2 & 0 & 2 \\ 0 & 2 & 0 & 2 & 0 \\ 1 & 0 & 0 & 0 & 1 \end{bmatrix} * X_{t-1} + \begin{bmatrix} .5 & -5 & 0 & 0 \\ 0 & .5 & 0 & -5 \\ 0 & 0 & -1 & -5 \\ 0 & -5 & 0 & -1 \\ -5 & 0 & -5 & 0 \end{bmatrix} * I_t)$$

where $t$ is time, $X_t$ is the 5-dimensional state vector, $I_t$ is the 4-dimensional input vector using 1-hot encoding of $a = [1, 0, 0, 0]; b = [0, 1, 0, 0]; A = [0, 0, 1, 0], B = [0, 0, 0, 1]$. The simple trick is to use the input weights to turn on or off the counting. For example, the dimension-5 state variable is turned off when input is $a$ or $A$, but then turned on when $b$ is input, at which time it copies the last $a$ value and holds on to it. It is then easy to add Boolean output decision functions that keep predictions linearly separable.

However, other solutions are possible. Rather than store the $a$ count one could keep counting up in dimension-1 for $b$ input and then cancel it by counting down for $B$ input. The questions that arise are: Can an RNN implement a solution that generalizes? What kind of store and copy mechanism does an RNN discover?

## 2.3 TRAINING DATA & RESULTS

The training set consists of 68 possible strings of total length $\leq 25$, which means a maximum of $n + m = 12$, or 12 symbols in the first half, 12 symbols in the second half, and 1 end symbol [2]. The complete training set has more short strings so that the network does not disregard the transitions at the end of the string or at the end of the $b...B$ subsequence. The network consists of 5 input, 5 hidden, 5 output units, with a bias node. The hidden and recurrent units are updated in the same time step as the input is presented. The recurrent layer activations are input on the next time step. The weight updates are performed using back-propagation thru time training with error injected at each time step backward for 24 time steps for each input.

We found that about half our simulations learn to make predictions for transitions, and most will have few generalizations on longer strings not seen in the training set. However, no network learned the complete training set perfectly. The best network was trained for 250K sweeps (1 per character) with a learning parameter of .001, and 136K more sweeps with .0001, for a total of about 51K strings. The network made 28 total prediction errors on 28

different strings in the test set of 68 possible strings seen in training. All of these errors were isolated to 3 situations: when the number of $a$ input $= 2$ or $4$ the error occurred at the $B$-to-$A$ transition, when the number of $a$ input $= 1$, for $m > 2$, the error occurred as an early $A$-to-end transition.

Importantly, the network made correct predictions on many strings longer than seen in training, e.g. strings that have total length $> 25$ (or $n + m > 12$). It counted longer strings of $a..A$s with or without embedded $b..B$s; such as: $w = a^{13}$; $w = a^{13}b^2$; $w = a^n b^7$, $n = 6, 7$ or $8$ (recall that $w$ is the first half of the string). It also generalized to count longer subsequences of $b..B$s with or without more $a..A$s; such as $w = a^5 b^n$, where $n = 8, 9, 10, 11, 12$. The longest string it processed correctly was $w = a^9 b^9$, which is 12 more characters than seen during training. The network learned to store the count for $a^9$ for up to $9 b$s, even though the longest example it had seen in training had only $3 b$s - clearly it's doing something right.

## 2.4  NETWORK EVALUATION

Our evaluation will focus on how the best network counts, copies, and stores information. We use a mix of graphical analysis and linear system analysis, to piece together a global picture of how phase space trajectories hold informational states. The linear system analysis consists of investigating the local behaviour of the Jacobian at fixed points under each input condition separately. We refer to $F_a$ as the autonomous system under $a$ input condition and similarly for $F_b$, $F_A$, and $F_B$.

The most salient aspect to the solution is that the network divides up the processing along different dimensions in space. By inspection we note that hidden unit1 (HU1) takes on low values for the first half of the string and high values for the second half, which helps keep the processing linearly separable. Therefore in the graphical analysis of the RNN we can set HU1 to a constant.

First, we can evaluate how the network counts the $b..B$ subsequences. Again, by inspection the network uses dimensions HU3,HU4. The graphical analysis in Figure 1a and Figure 1b plots the activity of HU3xHU4. It shows how the network counts the right number of $B$s and then makes a transition to predict the first $A$. The dominant eigenvalues at the $F_b$ attracting point and $F_B$ saddle point are inversely proportional, which indicates that the contraction rate to and expansion rate away from the fixed points are inversely matched. The $F_B$ system expands out to a periodic-2 fixed point in HU3xHU4 subspace, and the unstable eigenvector corresponding to the one unstable eigenvalue has components only in HU3,HU4. In Figure 2 we plot the vector field that describes the flow in phase space for the composite $F_B^2$, which shows the direction where the system contracts along the stable manifold, and expands on the unstable manifold. One can see that the nature of the transition after the last $b$ to the first $B$ is to place the state vector close to saddle point for $F_B$ so that the number of expansion steps matches the number of the $F_b$ contraction steps. In this way the $b$ count is copied over to a different region of phase space.

Now we evaluate how the network counts $a...A$, first without any $b...B$ embedding. Since the output unit for the end symbol has very high weight values for HU2, and the $F_a$ system has little activity in HU4, we note that $a$ is processed in HU2xHU3xHU5. The trajectories in Figure 3 show a plot of $a^{13} A^{13}$ that properly predicts all $A$s as well as the transition at the end. Furthermore, the dominant eigenvalues for the $F_a$ attracting point and the $F_A$ saddle point are nearly inversely proportional and the $F_A$ system expands to a periodic-2 fixed point in 4-dimensions (HU1 is constant, whereas the other HU values are periodic). The $F_a$ eigenvectors have strong-moderate components in dimensions HU2, HU3, HU5; and likewise in HU2, HU3, HU4, HU5 for $F_A$.

The much harder question is: How does the network maintain the information about the count of $a$s that were input while it is processing the $b..B$ subsequence? Inspection shows

that after processing $a^n$ the activation values are not directly copied over any HU values, nor do they latch any HU values that indicate how many $a$s were processed. Instead, the last state value after the last $a$ affects the dynamics for $b...B$ in such a way that clusters the last state value after the last $B$, but only in HU3xHU4 space (since the other HU dimensions were unchanging throughout $b...B$ processing).

We show in Figure 4 the clusters for state variables in HU3xHU4 space after processing $a^n b^m B^m$, where $n = 2, 3, 4, 5 or 6; m = 1..10$. The graph shows that the information about how many $a$'s occurred is "stored" in the HU3xHU4 region where points are clustered. Figure 4 includes the dividing line from Figure 1b for the predict $A$ region. The network does not predict the $B$-to-$A$ transition after $a^4$ or $a^2$ because it ends up on the wrong side of the dividing line of Figure 1b, but the network in these cases still predicts the $A$-to-end transition. We see that if the network did not oscillate around the $F_B$ saddle point while exanding then the trajectory would end up correctly on one side of the decision plane.

It is important to see that the clusters themselves in Figure 4 are on a contracting trajectory toward a fixed point, which stores information about increasing number of $a$s when matched by an expansion of the $F_A$ system. For example, the state values after $a^5 AA$ and $a^5 b^m B^m AA, m = 2..10$ have a total hamming distance for all 5 dimensions that ranged from .070 to .079. Also, the fixed point for the $F_a$ system, the estimated fixed point for the composite $F_B^m \circ F_b^m \circ F_a^n$, and the saddle point of the $F_A$ system are colinear [3]. in all the relevant counting dimensions: 2,3,4, and 5. In other words, the $F_A$ system contracts the different coordinate points, one for $a^n$ and one for $a^n b^m B^m$, towards the saddle point to nearly the same location in phase space, treating those points as having the same information. Unfortunately, this is a contraction occuring through a 4 dimensional subspace which we cannot easily show graphically.

# 3 CONCLUSION

In conclusion, we have shown that an RNN can develop a symbol-sensitive counting solution for a simple palindrome. In fact, this solution is not a stack but consists of non-independent counters that use dynamics to visit different regions at appropriate times. Furthermore, an RNN can implement counting solutions for a prediction task that are functionally similar to that prescribed by analog computation theory, but the store and copy functions rely on distance in phase space to implicitly affect other trajectories.

**Acknowledgements**

This research was funded by the UCSD, Center for Research in Language Training Grant to Paul Rodriguez, and a grant from the Australian Research Council to Janet Wiles.

**References**

Casey, M. (1996) The Dynamics of Discrete-Time Computation, With Application to Recurrent Neural Networks and Finite State Machine Extraction. Neural Computation, 8.

Elman, J.L. (1990) Finding Structure in Time. Cognitive Science, 14, 179-211.

Maass, W. , Orponen, P. (1997) On the Effect of Analog Noise in Discrete-Time Analog Computations. Proceedings Neural Information Processing Systems, 1996.

Moore, C. (1996) Dynamical Recognizers: Real-Time Language Recognition by Analog Computation. Santa Fe InstituteWorking Paper 96-05-023.

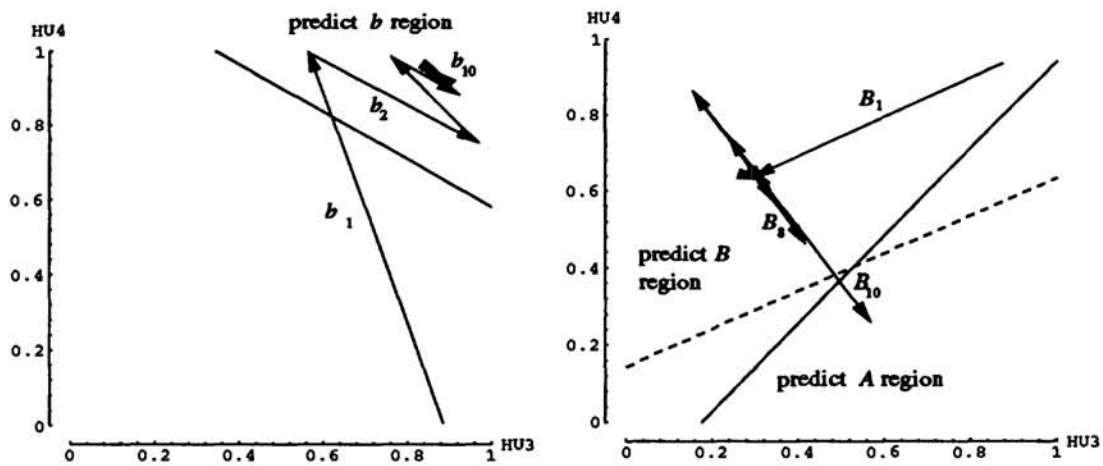

Figure 1: 1a)Trajectory of $b^{10}$ (after $a^5$) in HU3xHU4. Each arrow represents a trajectory step:the base is a state vector at time $t$, the head is a state at time $t + 1$. The first $b$ trajectory step has a base near (.9,.05), which is the previous state from the last $a$. The output node b is > .5 above the dividing line. 1b) Trajectory of $B^{10}$ (after $a^5b^{10}$) in HU3xHU4. The output node B is > .5 above the dashed dividing line, and the output node A is > .5 below the solid dividing line. The system crosses the line on the last B step, hence it predicts the $B$-to-$A$ transition.

Pollack, J.B. (1991) The Induction of Dynamical Recognizers. Machine Learning, 7, 227-252.

Siegelmann, H.(1993) Foundations of Recurrent Neural Networks. Ph.D. dissertation, unpublished. New Brunswick Rutgers, The State University of New Jersey.

Wiles, J., Elman, J. (1995) Counting Without a Counter: A Case Study in Activation Dynamics. Proceedings of the Seventeenth Annual Conference of the Cognitive Science Society. Hillsdale, N.J.: Lawrence Erlbaum Associates.

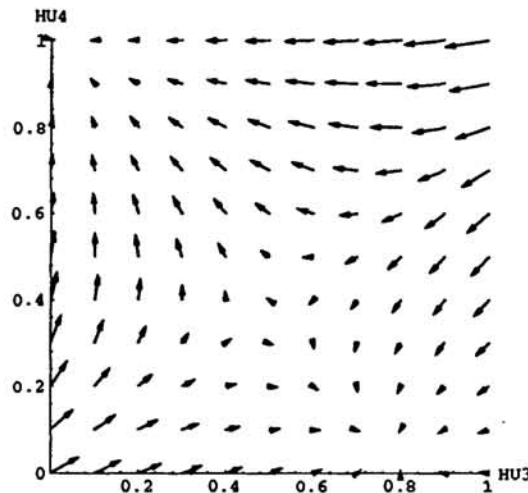

Figure 2: Vector field that describes the flow of $F_B^2$ projected onto HU3xHU4. The graph shows a saddle point near (.5,.5)and a periodic-2 attracting point.

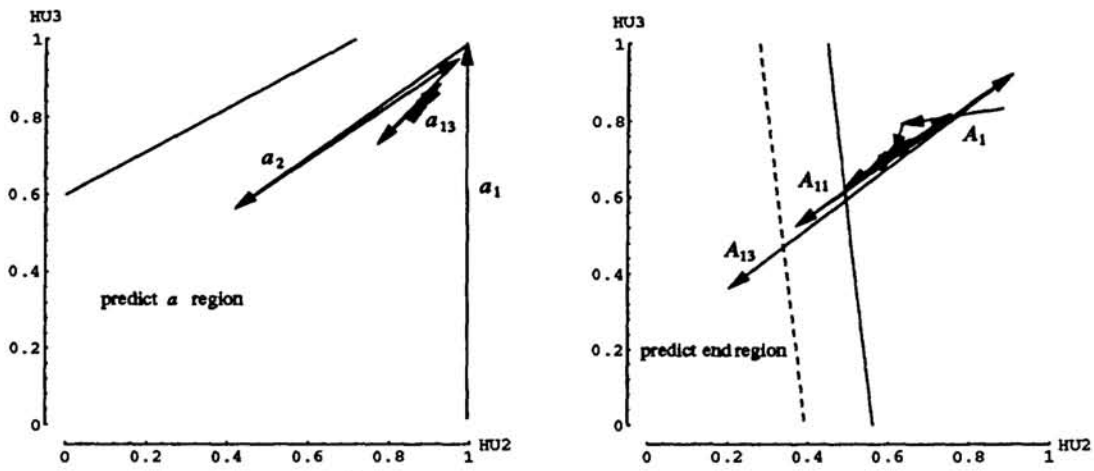

Figure 3: 3a) Trajectory of $a^{13}$ projected onto HU2xHU3. The output node a is > .5 below and right of dividing line. The projection for HU2xHU5 is very similar. 3b) Trajectory of $A^{13}$(after $a^{13}$) projected onto HU2xHU3. The output node for the end symbol is > .5 on the 13th trajectory step left of the solid dividing line, and it is > .5 on the 11th step left of the dashed dividing line (the hyperplane projection must use values at the appropriate time steps), hence the system predicts the $A$-to-end transition. The graph for HU2xHU5 and HU2xHU4 is very similar.

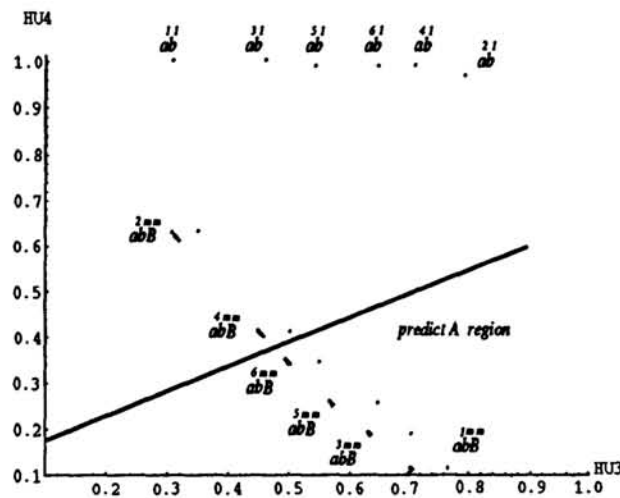

Figure 4: Clusters of last state values $a^n b^m B^m$, $m > 1$, projected onto HU3xHU4. Notice that for increasing $n$ the system oscillates toward an attracting point of the system $F_B^m \circ F_b^m \circ F_a^n$.

## Footnotes

[1]These can be expanded relatively easily to include more symbols, different symbol representations, harder palindrome sequences, or different kind of decision planes.

[2]We removed training strings $w = a^n b$, for $n > 1$; it turns out that the network interpolates on the $B$-to-$A$ transition for these. Also, we added an end symbol to help reset the system to a consistent starting value.

[3] Relative to the saddle point, the vector for one fixed point, multiplied by a constant had the same value(to within .05) in each of 4 dimensions as the vector for the other fixed point
